# PROGRAMMABLE ANALOG PULSE-FIRING NEURAL NETWORKS

Alan F. Murray
Dept. of Elec. Eng.,
University of Edinburgh,
Mayfield Road,
Edinburgh, EH9 3JL
United Kingdom.

Alister Hamilton
Dept. of Elec. Eng.,
University of Edinburgh,
Mayfield Road,
Edinburgh, EH9 3JL
United Kingdom.

Lionel Tarassenko
Dept. of Eng. Science,
University of Oxford,
Parks Road,
Oxford, OX1 3PJ
United Kingdom.

## ABSTRACT

We describe pulse - stream firing integrated circuits that implement asynchronous analog neural networks. Synaptic weights are stored dynamically, and weighting uses time-division of the neural pulses from a signalling neuron to a receiving neuron. MOS transistors in their "ON" state act as variable resistors to control a capacitive discharge, and time-division is thus achieved by a small synapse circuit cell. The VLSI chip set design uses $2.5\mu$m CMOS technology.

## INTRODUCTION

Neural network implementations fall into two broad classes - digital [1, 2] and analog (e.g. [3, 4]). The strengths of a digital approach include the ability to use well-proven design techniques, high noise immunity, and the ability to implement programmable networks. However digital circuits are synchronous, while biological neural networks are asynchronous. Furthermore, digital multipliers occupy large areas of silicon. Analog networks offer asynchronous behaviour, smooth neural activation and (potentially) small circuit elements. On the debit side, however, noise immunity is low, arbitrary high precision is not possible; and no reliable "mainstream" analog nonvolatile memory technology exists.

Many analog VLSI implementations are nonprogrammable, and therefore have fixed functionality. For instance, subthreshold MOS devices have been used to mimic the nonlinearities of neural behaviour, in implementing Hopfield style nets [3] , associative memory [5] , visual processing functions [6] , and auditory processing [7]. Electron-beam programmable resistive interconnects have been used to represent synaptic weights between more conventional operational-amplifier neurons [8, 4].

We describe *programmable analog pulse-firing neural networks* that use on-chip dynamic *analog* storage capacitors to store synaptic weights, currently

refreshed from an external RAM via a Digital →Analog converter.

## PULSE-FIRING NEURAL NETWORKS

A pulse-firing neuron, $i$ is a circuit which signals its state, $V_i$ by generating a stream of 0→5V pulses on its output. The pulse rate $R_i$ varies from 0 when neuron $i$ is OFF to $R_i(\text{max})$ when neuron $i$ is fully ON. Switching between the OFF and ON states is a smooth transition in output pulse rate between these lower and upper limits. In a previous system, outlined below, the *synapse* allows a proportion of complete presynaptic neural pulses $V_j$ to be added (electrically OR-ed) to its output. A synaptic "gating" function, determined by $T_{ij}$, allowed bursts of complete pulses through the synapse. Moving down a column of synapses, therefore, we see an ever more crowded asynchronous mass of pulses, representing the aggregated activity of the receiving neuron. In the system that forms the substance of this paper, a proportion (determined by $T_{ij}$) of *each presynaptic pulse* is passed to the postsynaptic summation.

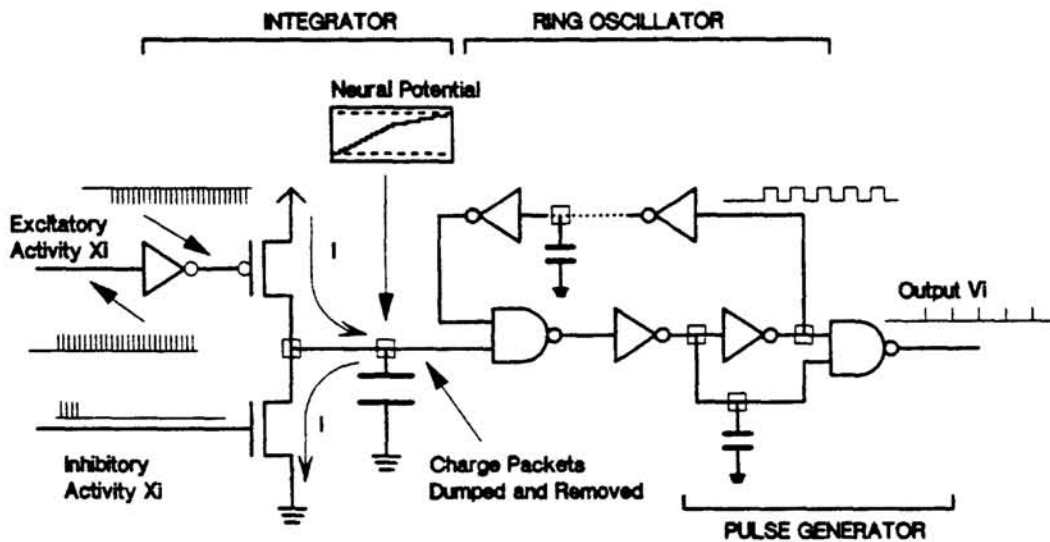

**Figure 1.** Neuron Circuit

## NEURON CIRCUIT

Figure 1 shows a CMOS implementation of the pulse-firing neuron function in a system where excitatory and inhibitory pulses are accumulated on separate channels. The output stage of the neuron consists of a "ring oscillator" - a feedback circuit containing an odd number of logic inversions, with the loop broken by a NAND gate, controlled by a smoothly varying voltage representing the neuron's total activity,

$$x_i = \sum_{j=0}^{j=n-1} T_{ij} V_j$$

This activity is increased or decreased by the dumping or removal of charge packets from the "integrator" circuit. The arrival of an excitatory pulse dumps charge, while an inhibitory pulse removes it. Figure 2 shows a device level (SPICE) simulation of the neuron circuit. A strong excitatory input causes the neural potential to rise in steps and the neuron turns ON. Subsequent inhibitory pulses remove charge packets from the integrating capacitor at a higher rate, driving the neuron potential down and switching the neuron OFF.

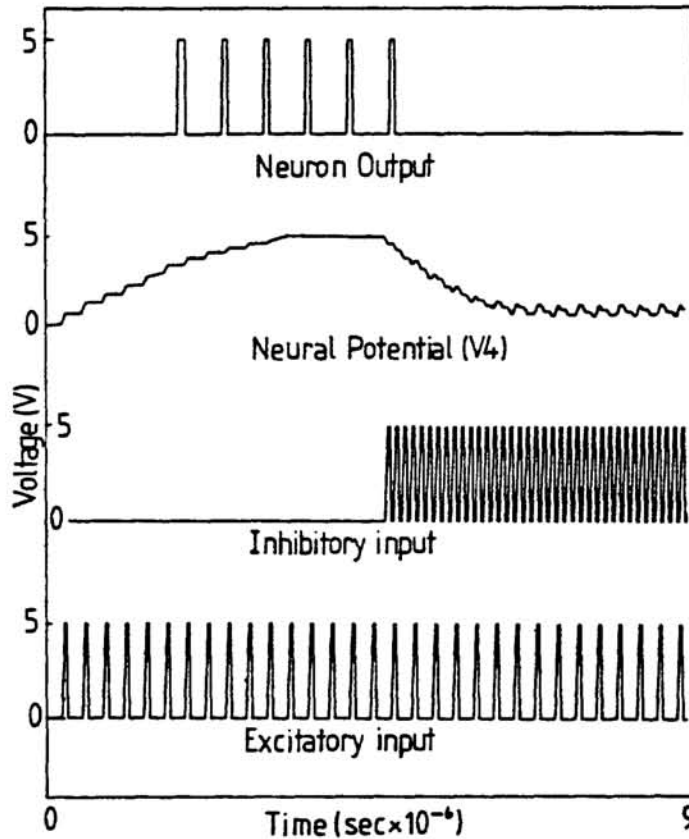

**Figure 2.** SPICE Simulation of Neuron

## SYNAPSE CIRCUIT - USING CHOPPING CLOCKS

In an earlier implementation, "chopping clocks" were introduced - synchronous to one another, but asynchronous to the neural firing. One bit of the (digitally stored) weight $T_{ij}$ indicates its sign, and each other bit of precision is represented by a chopping clock. The clocks are non-overlapping, the MSB clock is high for ½ of the time, the next for ¼ of the time, etc. These clocks are used to *gate* bursts of pulses such that a fraction $T_{ij}$ of the pulses are passed from the input of the synapse to either the excitatory or inhibitory output channel.

## CHOPPING CLOCK SYSTEM - PROBLEMS

A custom VLSI synaptic array has been constructed [9] with the neural function realised in discrete SSI to allow flexibility in the choice of time constants. The technique has proven successful, but suffers from a number of problems:-

- Digital gating ("using chopping clocks") is clumsy
- Excitation and Inhibition on separate lines - bulky
- Synapse complicated and of large area
- < 100 **synapses** per chip
- < 10 **neurons** per chip

In order to overcome these problems we have devised an alternative arithmetic technique that modulates individual pulse widths and uses analog dynamic weight storage. This results in a much smaller synapse.

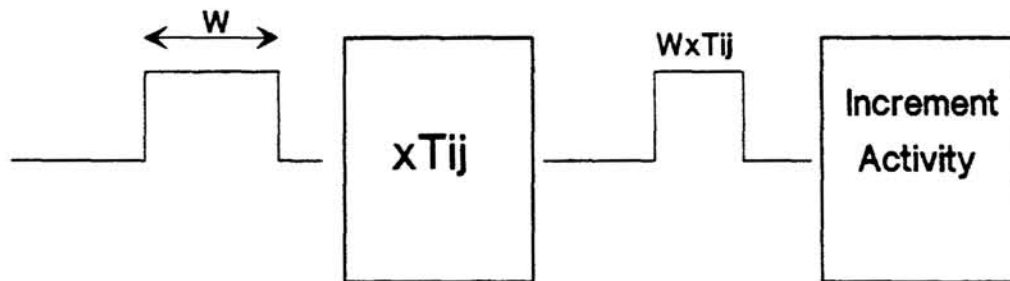

**Figure 3.** Pulse Multiplication

## SYNAPSE CIRCUIT - PULSE MULTIPLICATION

The principle of operation of the new synapse is illustrated in Figure 3. Each presynaptic pulse of width W is modulated by the synaptic weight $T_{ij}$ such that the resulting postsynaptic pulse width is

$$W.T_{ij}$$

This is achieved by using an analog voltage to modulate a capacitive discharge as illustrated in Figure 4. The presynaptic pulse enters a CMOS inverter whose positive supply voltage (Vdd) is controlled by $T_{ij}$. The capacitor is nominally charged to Vdd, but begins to discharge at a constant rate when the input pulse arrives. When the voltage on the capacitor falls below the threshold of the following inverter, the synapse output goes high. At the end of the presynaptic pulse the capacitor recharges rapidly and the synapse output goes low, having output a pulse of length $W.T_{ij}$. The circuit is now

ready for the next presynaptic pulse. This mechanism gives a linear relationship between multiplier W and inverter supply voltage, Vdd.

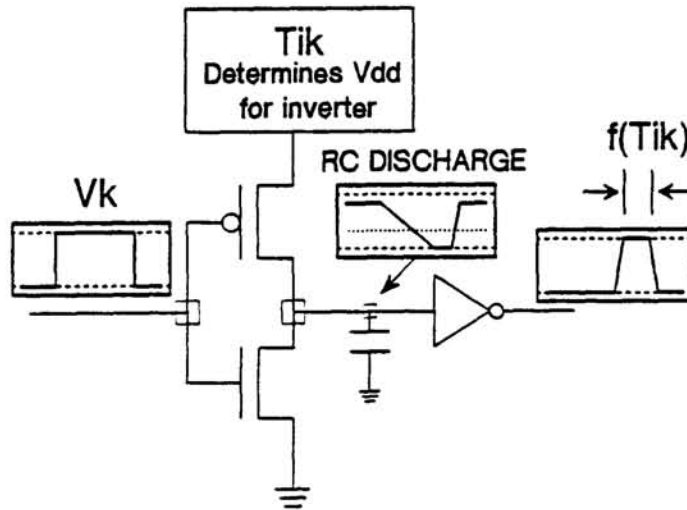

**Figure 4.** Improved Synapse Circuit

## FULL SYNAPSE

Synaptic weight storage is achieved using dynamic *analog* storage capacitors refreshed from off-chip RAM via a Digital→Analog converter. A CMOS active-resistor inverter is used as a buffer to isolate the storage capacitor from the multiplier circuit as shown in the circuit diagram of a full synapse in Figure 5.

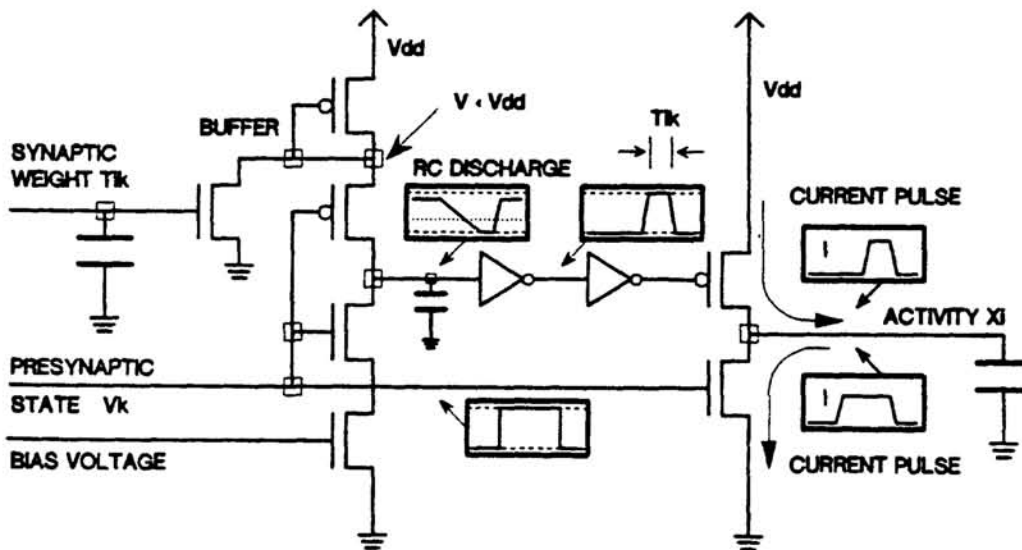

**Figure 5.** Full Synapse Circuit

A capacitor distributed over a column of synaptic outputs stores neural activity, $x_i$, as an analog voltage. The range over which the synapse voltage → pulse time multiplier relationship is linear is shown in Figure 6. This wide

($\approx 2$V) range may be used to implement inhibition and excitation in a single synapse, by "splitting" the range such that the lower volt (1-2V) represents inhibition, and the upper volt (2-3V) excitation. Each presynaptic pulse removes a packet of charge from the activity capacitor while each postsynaptic pulse adds charge at twice the rate. In this way, a synaptic weight voltage of 2V, giving a pulse length multiplier of ½, gives no net change in neuron activity $x_i$. The synaptic weight voltage range $1 \rightarrow 2$V therefore gives a net reduction in neuron activity and is used to represent inhibition, the range $2 \rightarrow 3$V gives a net increase in neuron activity and is used to represent excitation.

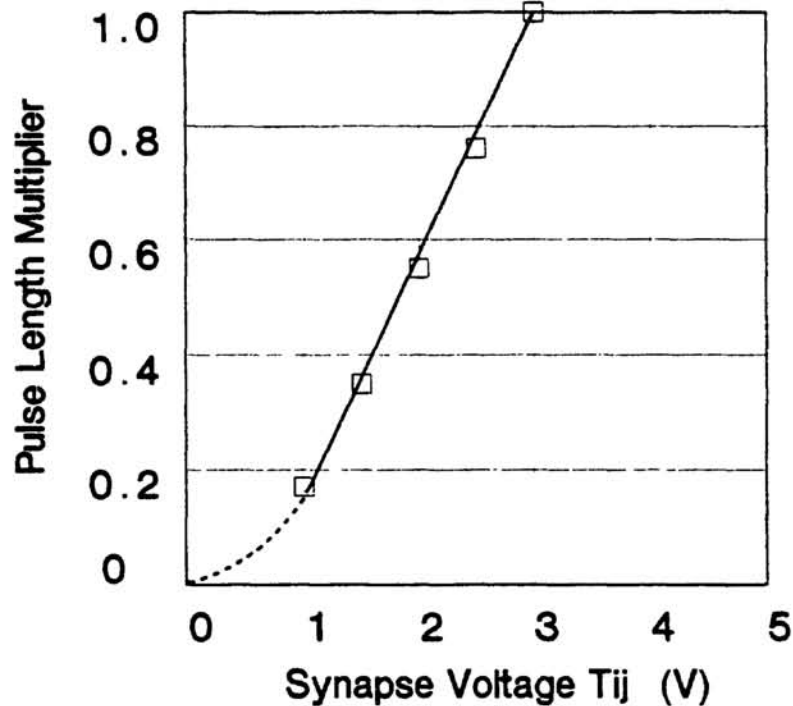

**Figure 6.** Multiplier Linearity

The resulting synapse circuit implements excitation and inhibition in 11 transistors per synapse. It is estimated that this technique will yield more than 100 fully programmable neurons per chip.

## FURTHER WORK

There is still much work to be done to refine the circuit of Figure 5 to optimise (for instance) the mark-space ratio of the pulse firing and the effect of pulse overlap, and to minimise the power consumption. This will involve the creation of a custom pulse-stream simulator, implemented directly as code, to allow these parameters to be studied in detail in a way that probing an actual chip does not allow. Finally, as Hebbian- (and modified Hebbian - for instance [10]) learning schemes only require a synapse to "know" the presynaptic and postsynaptic states, we are able to implement it on-chip at little cost, as the chip topology makes both of these signals available available to the synapse locally. This work introduces as many exciting possibilities for truly autonomous systems as it does potential problems!

## Acknowledgements

The authors acknowledge the support of the Science and Engineering Research Council (UK) in the execution of this work.

## References

1.  A. F. Murray, A. V. W. Smith, and Z. F. Butler, "Bit - Serial Neural Networks," *Neural Information Processing Systems (Proc. 1987 NIPS Conference)*, p. 573, 1987.

2.  S. C. J. Garth, "A Chipset for High Speed Simulation of Neural Network Systems," *IEEE Conference on Neural Networks, San Diego*, vol. 3, pp. 443 - 452, 1987.

3.  M. A. Sivilotti, M. R. Emerling, and C. A. Mead, "VLSI Architectures for Implementation of Neural Networks," *Proc. AIP Conference on Neural Networks for Computing, Snowbird*, pp. 408 - 413, 1986.

4.  H. P. Graf, L. D. Jackel, R. E. Howard, B. Straughn, J. S. Denker, W. Hubbard, D. M. Tennant, and D. Schwartz, "VLSI Implementation of a Neural Network Memory with Several Hundreds of Neurons," *Proc. AIP Conference on Neural Networks for Computing, Snowbird*, pp. 182 - 187, 1986.

5.  M. Sivilotti, M. R. Emerling, and C. A. Mead, "A Novel Associative Memory Implemented Using Collective Computation," *Chapel Hill Conf. on VLSI*, pp. 329 - 342, 1985.

6.  M. A. Sivilotti, M. A. Mahowald, and C. A. Mead, "Real - Time Visual Computations Using Analog CMOS Processing Arrays," *Stanford VLSI Confeence*, pp. 295-312, 1987.

7.  C. A. Mead, in *Analog VLSI and Neural Systems*, Addison-Wesley, 1988.

8.  W. Hubbard, D. Schwartz, J. S. Denker, H. P. Graf, R. E. Howard, L. D. Jackel, B. Straughn, and D. M. Tennant, "Electronic Neural Networks," *Proc. AIP Conference on Neural Networks for Computing, Snowbird*, pp. 227 - 234, 1986.

9.  A. F. Murray, A. V. W. Smith, and L. Tarassenko, "Fully-Programmable Analogue VLSI Devices for the Implementation of Neural Networks," *Int. Workshop on VLSI for Artificial Intelligence*, 1988.

10. S. Grossberg, "Some Physiological and Biochemical Consequences of Psychological Postulates," *Proc. Natl. Acad. Sci. USA*, vol. 60, pp. 758 - 765, 1968.